# Who Does What? A Novel Algorithm to Determine Function Localization

**Ranit Aharonov-Barki**
Interdisciplinary Center for Neural Computation
The Hebrew University, Jerusalem 91904, Israel
*ranit@alice.nc.huji.ac.il*

**Isaac Meilijson and Eytan Ruppin**
School of Mathematical Sciences
Tel-Aviv University, Tel-Aviv, Israel
*isaco@math.tau.ac.il, ruppin@math.tau.ac.il*

## Abstract

We introduce a novel algorithm, termed PPA (Performance Prediction Algorithm), that quantitatively measures the contributions of elements of a neural system to the tasks it performs. The algorithm identifies the neurons or areas which participate in a cognitive or behavioral task, given data about performance decrease in a small set of lesions. It also allows the accurate prediction of performances due to multi-element lesions. The effectiveness of the new algorithm is demonstrated in two models of recurrent neural networks *with complex interactions* among the elements. The algorithm is scalable and applicable to the analysis of large neural networks. Given the recent advances in *reversible inactivation techniques*, it has the potential to significantly contribute to the understanding of the organization of biological nervous systems, and to shed light on the long-lasting debate about local versus distributed computation in the brain.

## 1 Introduction

Even simple nervous systems are capable of performing multiple and unrelated tasks, often in parallel. Each task recruits some elements of the system (be it single neurons or cortical areas), and often the same element participates in several tasks. This poses a difficult challenge when one attempts to identify the roles of the network elements, and to assess their contributions to the different tasks. Assessing the importance of single neurons or cortical areas to specific tasks is usually achieved either by assessing the deficit in performance after a lesion of a specific area, or by recording the activity during behavior, assuming that areas which deviate from baseline activity are more important for the task performed. These *classical methods suffer from two fundamental flaws:* First, they do not take into account the probable case that there are complex interactions among elements in the system. E.g., if two neurons have a high degree of redundancy, lesioning of either one alone will not reveal its influence. Second, they are qualitative measures, lacking quantitative predictions.

Moreover, the very nature of the contribution of a neural element is quite elusive and ill defined. In this paper **we propose both a rigorous, operative definition for the neuron's contribution and a novel algorithm to measure it.**

Identifying the contributions of elements of a system to varying tasks is often used as a basis for claims concerning the degree of the distribution of computation in that system (e.g. [1]). The *distributed representation* approach hypothesizes that computation emerges from the interaction between many simple elements, and is supported by evidence that many elements are important in a given task [2, 3, 4]. The *local representation* hypothesis suggests that activity in single neurons represents specific concepts (the *grandmother cell* notion) or performs specific computations (see [5]). This question of distributed versus localized computation in nervous systems is fundamental and has attracted ample attention. However there seems to be a lack of a unifying definition for these terms [5]. The ability of the new algorithm suggested here, to quantify the contribution of elements to tasks, allows us to deduce both the distribution of the different tasks in the network and the degree of specialization of each neuron.

We applied the Performance Prediction Algorithm (PPA) to two models of recurrent neural networks: The first model is a network hand-crafted to exhibit redundancy, feedback and modulatory effects. The second consists of evolved neurocontrollers for behaving autonomous agents [6]. In both cases the algorithm results in measures which are highly consistent with what is qualitatively known a-priori about the models. The fact that these are recurrent networks, and not simple feed-forward ones, suggests that the algorithm can be used in many classes of neural systems which pose a difficult challenge for existing analysis tools. Moreover, the proposed algorithm is scalable and applicable to the analysis of large neural networks. It can thus make a major contribution to studying the organization of tasks in biological nervous systems as well as to the long-debated issue of local versus distributed computation in the brain.

## 2 Indices of Contribution, Localization and Specialization

### 2.1 The Contribution Matrix

Assume a network (either natural or artificial) of $N$ neurons performing a set of $P$ different functional tasks. For any given task, we would like to find the *contribution vector* $c = (c_1, ..., c_N)$, where $c_i$ is the contribution of neuron $i$ to the task in question. We suggest a rigorous and operative definition for this contribution vector, and propose an algorithm for its computation.

Suppose a set of neurons in the network is lesioned and the network then performs the specified task. The result of this experiment is described by the pair $< m, p_m >$ where $m$ is an incidence vector of length $N$, such that $m(i) = 0$ if neuron $i$ was lesioned and 1 if it was intact. $p_m$ is the *performance* of the network divided by the baseline case of a fully intact network.

Let the pair $< f, c >$, where $f$ is a smooth monotone non-decreasing[1] function and $c$ a normalized column vector such that $\sum_{i=1}^{N} |c_i| = 1$, be the pair which minimizes the following error function

$$E = \frac{1}{2^N} \sum_{\{m\}} [f(m \cdot c) - p_m]^2.$$

(1)

This $c$ will be taken as the *contribution vector* for the task tested, and the corresponding $f$ will be called its adjoint *performance prediction function*.

Given a configuration $m$ of lesioned and intact neurons, the predicted performance of the network is the sum of the contribution values of the intact neurons ($m \cdot c$), passed through the performance prediction function $f$. The contribution vector $c$ accompanied by $f$ is optimal in the sense that this predicted value minimizes the Mean Square Error relative to the real performance, over all possible lesion configurations.

The computation of the contribution vectors is done separately for each task, using some small subset of all the $2^N$ possible lesioning configurations. The training error $E_t$ is defined as in equation 1 but only averaging on the configurations present in the training set.

**The Performance Prediction Algorithm (PPA):**

- **Step 1:** Choose an initial normalized contribution vector $c$ for the task. If there is no bias for a special initial choice, set all entries to random values.

  Repeat steps 2 and 3 until the error $E_t$ converges or a maximal number of steps has been reached:

- **Step 2: Compute $f$.** Given the current $c$, perform isotonic regression [7] on the pairs $< m \cdot c, p_m >$ in the training set. Use a smoothing spline [8] on the result of the regression to obtain the new $f$.

- **Step 3: Compute $c$.** Using the current $f$ compute new $c$ values by training a perceptron with input $m$, weights $c$ and transfer function $f$. The output of the perceptron is exactly $f(m \cdot c)$, and the target output is $p_m$. Hence training the perceptron results in finding a new vector $c$, that given the current function $f$, minimizes the error $E_t$ on the training set. Finally re-normalize $c$.

The output of the algorithm is thus a contribution value for every neuron, accompanied by a function, such that given any configuration of lesioned neurons, one can predict with high confidence the performance of the damaged network. **Thus, the algorithm achieves two important goals:** a) It identifies automatically the neurons or areas which participate in a cognitive or behavioral task. b) The function $f$ predicts the result of multiple lesions, allowing for non linear combinations of the effects of single lesions [2].

The application of the PPA to all tasks defines a *contribution matrix* $C$, whose $k^{th}$ column ($k = 1...P$) is the contribution vector computed using the above algorithm for task $k$, i.e. $C_{ik}$ is the contribution of neuron $i$ to task $k$.

## 2.2 Localization and Specialization

Introducing the contribution matrix allows us to approach issues relating to the distribution of computation in a network in a quantitative manner. Here we suggest quantitative measures for localization of function and specialization of neurons.

If a task is completely distributed in the network, the contributions of all neurons to that task should be identical (full *equipotentiality* [2]). Thus, we define the *localization* $L_k$ of task $k$ as a deviation from equipotentiality. Formally, $L_k$ is the standard deviation of column $k$ of the contribution matrix divided by the maximal possible standard deviation.

$$L_k = \frac{std(C_{*k})}{\sqrt{(N-1)/N^2}}. \tag{2}$$

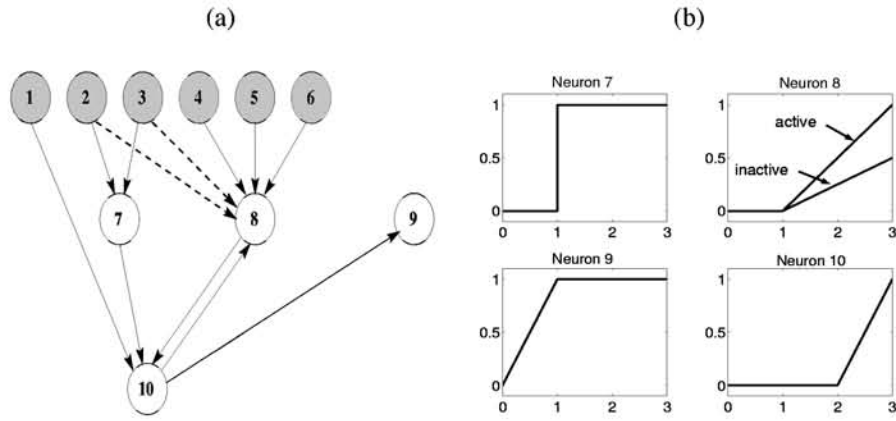

Figure 1: **Hand-crafted neural network:** a) Architecture of the network. Solid lines are weights, all of strength 1. Dashed lines indicate modulatory effects. Neurons 1 through 6 are spontaneously active (activity equals 1) under normal conditions. The performance of the network is taken to be the activity of neuron 10. b) The activation functions of the non-spontaneous neurons. The x axis is the input field and the y axis is the resulting activity of the neuron. Neuron 8 has two activation functions. If both neurons 2 and 3 are switched on they activate a modulating effect on neuron 8 which switches its activation function from the inactive case to the active case.

Note that $L_k$ is in the range $[0, 1]$ where $L_k = 0$ indicates full distribution and $L_k = 1$ indicates localization of the task to one neuron alone. The degree of localization of function in the whole network, $L$, is the simple average of $L_k$ over all tasks. Similarly, if neuron $i$ is highly specialized for a certain task, $C_{i*}$ will deviate strongly from a uniform distribution, and thus we define $S_i$, the *specialization* of neuron $i$ as

$$S_i = \frac{std(|C_{i*}|)}{\sqrt{(P-1)/P^2}}.$$

(3)

## 3  Results

We tested the proposed index on two types of recurrent networks. We chose to study recurrent networks because they pose an especially difficult challenge, as the output units also participate in the computation, and in general complex interactions among elements may arise[3]. We begin with a hand-crafted example containing redundancy, feedback and modulation, and continue with networks that emerge from an evolutionary process. The evolved networks are not hand-crafted but rather their structure emerges as an outcome of the selection pressure to successfully perform the tasks defined. Thus, we have no prior knowledge about their structure, yet they are tractable models to investigate.

### 3.1  Hand-Crafted Example

Figure 1 depicts a neural network we designed to include potential pitfalls for analysis procedures aimed at identifying important neurons of the system (see details in the caption). Figure 2(a) shows the contribution values computed by three methods applied to this network. The first estimation was computed as the correlation between the activity of the

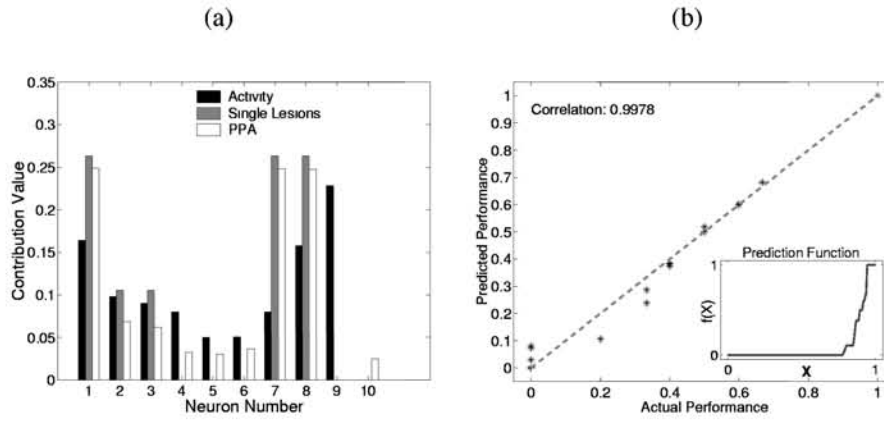

(a)                          (b)

Figure 2: **Results of the PPA:** a) Contribution values obtained using three methods: The correlation of activity to performance, single neuron lesions, and the PPA. b) Predicted versus actual performance using $c$ and its adjoint performance prediction function $f$ obtained by the PPA. *Insert:* The shape of $f$.

neuron and the performance of the network[4]. To allow for comparison between methods these values were normalized to give a sum of 1. The second estimation was computed as the decrease in performance due to lesioning of single neurons. Finally, we used the PPA, training on a set of 64 examples. Note that as expected the activity correlation method assigns a high contribution value to neuron 9, even though it actually has no significance in determining the performance. Single lesions fail to detect the significance of neurons involved in redundant interactions (neurons $4 - 6$). The PPA successfully identifies the underlying importance of all neurons in the network, even the subtle significance of the feedback from neuron 10. We used a small training set (64 out of $2^{10}$ configurations) containing lesions of either small (up to 20% chance for each neuron to be lesioned) or large (more than 90% chance of lesioning) degree. Convergence was achieved after 10 iterations.

As opposed to the two other methods, the PPA not only identifies and quantifies the significance of elements in the network, but also allows for the prediction of performances from multi-element lesions, even if they were absent from the training set. The predicted performance following a given configuration of lesioned neurons is given by $f(m \cdot c)$ as explained in section 2.1. Figure 2(b) depicts the predicted versus actual performances on a test set containing 230 configurations of varying degrees ($0 - 100\%$ chance of lesioning). The correlation between the predicted value and the actual one is 0.9978, corresponding to a mean prediction error of only 0.0007. In principle, the other methods do not give the possibility to predict the performance in any straightforward way, as is evident from the non-linear form of the performance prediction error (see insert of figure 2(b)). The shape of the performance prediction function depends on the organization of the network, and can vary widely between different models (results not shown here).

## 3.2   Evolved Neurocontrollers

Using evolutionary simulations we developed autonomous agents controlled by fully recurrent artificial neural networks. High performance levels were attained by agents performing simple life-like tasks of foraging and navigation. Using various analysis tools we found a common structure of a *command neuron* switching the dynamics of the network between

radically different behavioral modes [6]. Although the command neuron mechanism was a robust phenomenon, the evolved networks did differ in the role other neurons performed. When only limited sensory information was available, the command neuron relied on feedback from the motor units. In other cases no such feedback was needed, but other neurons performed some auxiliary computation on the sensory input. We applied the PPA to the evolved neurocontrollers in order to test its capabilities in a system on which we have previously obtained qualitative understanding, yet is still relatively complex.

Figure 3 depicts the contribution values of the neurons of three successful evolved neurocontrollers obtained using the PPA. Figure 3(a) corresponds to a neurocontroller of an agent equipped with a position sensor (see [6] for details), which does not require any feedback from the motor units. As can be seen these motor units indeed receive contribution values of near zero. Figures 3(b) and 3(c) correspond to neurocontrollers who strongly relied on motor feedback for their memory mechanism to function properly. The algorithm easily identifies their significance. In all three cases the command neuron receives high values as expected. The performance prediction capabilities are extremely high, giving correlations of 0.9999, 0.9922 and 0.9967 for the three neurocontrollers, on a test set containing 100 lesion configurations of mixed degrees ($0 - 100\%$ chance of lesioning). We also obtained the degree of localization of each network, as explained in section 2.2. The values are: 0.56, 0.35 and 0.47 for the networks depicted in figures 3(a) 3(b) and 3(c) respectively. These values are in good agreement with the qualitative descriptions of the networks we have obtained using classical neuroscience tools [6].

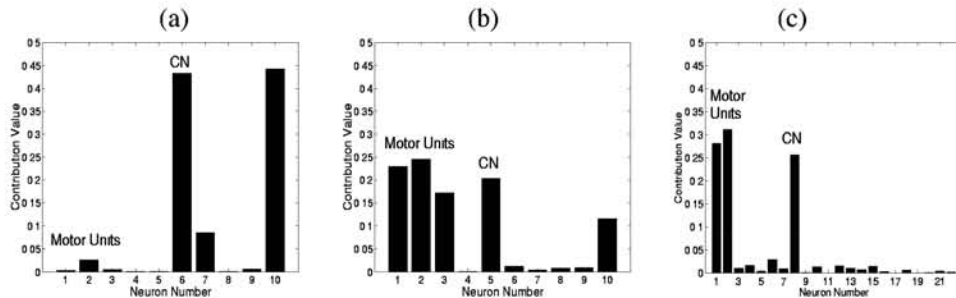

Figure 3: **Contribution values of neurons in three evolved neurocontrollers:** Neurons 1-4 are motor neurons. CN is the command neuron that emerged spontaneously in all evolutionary runs.

# 4   Discussion

We have introduced a novel algorithm termed PPA (Performance Prediction Algorithm) to measure the contribution of neurons to the tasks that a neural network performs. These contributions allowed us to quantitatively define an index of the degree of localization of function in the network, as well as for task-specialization of the neurons. The algorithm uses data from performance measures of the network when different sets of neurons are lesioned. Theoretically, pathological cases can be devised where very large training sets are needed for correct estimation. However it is expected that many cases are well-behaved and will demonstrate behaviors similar to the models we have used as test beds, i.e. that a relatively small subset suffices as a training set. It is predicted that larger training sets containing different degrees of damage will be needed to achieve good results for systems with higher redundancy and complex interactions. We are currently working on studying the nature of the training set needed to achieve satisfying results, as this in itself may reveal information on the types of interactions between elements in the system.

We have applied the algorithm to two types of artificial recurrent neural networks, and demonstrated that it results in agreement with our qualitative a-priori notions and with qualitative classical analysis methods. We have shown that estimation of the importance of system elements using simple activity measures and single lesions, may be misleading. The new PPA is more robust as it takes into account interactions of higher degrees. Moreover it serves as a powerful tool for *predicting* damage caused by multiple lesions, a feat that is difficult even when one can accurately estimate the contributions of single elements. The shape of the performance prediction function itself may also reveal important features of the organization of the network, e.g. its robustness to neuronal death.

The prediction capabilities of the algorithm can be used for regularization of recurrent networks. Regularization in feed-forward networks has been shown to improve performance significantly, and algorithms have been suggested for effective pruning [9]. However, networks with feedback (e.g. Elman-like networks) pose a difficult problem, as it is hard to determine which elements should be pruned. As the PPA can be applied on the level of single synapses as well as single neurons, it suggests a natural algorithm for effective regularization, pruning the elements by order of their contribution values.

Recently a large variety of *reversible inactivation techniques* (e.g. cooling) have emerged in neuroscience. These methods alleviate many of the problematic aspects of the classical lesion technique (ablation), enabling the acquisition of reliable data from multiple lesions of different configurations (for a review see [10]). It is most likely that a plethora of data will accumulate in the near future. The sensible integration of such data will require quantitative methods, to complement the available qualitative ones. The promising results achieved with artificial networks and the potential scalability of the PPA lead us to believe that it will prove extremely useful in obtaining insights into the organization of natural nervous systems.

### Acknowledgments

We greatly acknowledge the valuable contributions made by Ehud Lehrer, Hanoch Gutfreund and Tuvik Beker .

## Footnotes

[1]It is assumed that as more important elements are lesioned ($m \cdot c$ decreases), the performance ($p_m$) decreases, and hence the postulated monotonicity of $f$.

[2]The computation of $f$, involving a simple perceptron-based function approximation, implies the immediate applicability of the PPA for large networks, given well-behaved performance prediction functions.

[3]In order to single out the role of output units in the computation, lesioning was performed by decoupling their activity from the rest of the network and not by knocking them out completely.

[4]Neuron 10 was omitted in this method of analysis since it is by definition in full correlation with the performance.

# References

[1] J. Wu, L. B. Cohen, and C. X. Falk. Neuronal activity during different behaviors in aplysia: A distributed organiation? *Science*, 263:820–822, 1994.

[2] K. S. Lashley. *Brain Mechanisms in Intelligence*. University of Chicago Press, Chicago, 1929.

[3] J. L. McClelland, elhart D.E. Ru, and the PDP Research Group. *Parallel Distributed Processing: Explorations in the Microstructure of Cognition, Volume 2: Psychological and Biological Models*. MIT Press, Massachusetts, 1986.

[4] J. J. Hopfield. Neural networks and physical systems with emergent collective computational abilities. *Proceedings of the National Academy of Sciences, USA*, 79:2554–2558, 1982.

[5] S. Thorpe. Localized versus distributed representations. In M. A. Arbib, editor, *Handbook of Brain Theory and Neural Networks*. MIT Press, Massachusetts, 1995.

[6] R. Aharonov-Barki, T. Beker, and E. Ruppin. Emergence of memory-driven command neurons in evolved artificial agents. *Neural Computation*, In print.

[7] R. Barlow, D. Bartholemew, J. Bremner, and H. Brunk. *Statistical Inference Under Order Restrictions*. John Wiley, New York, 1972.

[8] G. D. Knott. Interpolating cubic splines. In National Science Foundation J. C. Cherniavsky, editor, *Progress in Computer Science and Applied Logic*. Birkhauser, 2000.

[9] R. Reed. Prunning algorithms - a survey. *IEEE Trans. on Neural Networks*, 4(5):740–747, 1993.

[10] S. G. Lomber. The advantages and limitations of permanent or reversible deactivation techniques in the assessment of neural function. *J. of Neuroscience Methods*, 86:109–117, 1999.
